# Efficient Inference in Phylogenetic InDel Trees

**Alexandre Bouchard-Côté**[†]     **Michael I. Jordan**[†‡]     **Dan Klein**[†]
Computer Science Division[†], Department of Statistics[‡]
University of California at Berkeley
Berkeley, CA 94720
{bouchard,jordan,klein}@cs.berkeley.edu

## Abstract

Accurate and efficient inference in evolutionary trees is a central problem in computational biology. While classical treatments have made unrealistic site independence assumptions, ignoring insertions and deletions, realistic approaches require tracking insertions and deletions along the phylogenetic tree—a challenging and unsolved computational problem. We propose a new *ancestry resampling* procedure for inference in evolutionary trees. We evaluate our method in two problem domains—multiple sequence alignment and reconstruction of ancestral sequences—and show substantial improvement over the current state of the art.

## 1   Introduction

Phylogenetic analysis plays a significant role in modern biological applications such as ancestral sequence reconstruction and multiple sequence alignment [1, 2, 3]. While insertions and deletions (InDels) of nucleotides or amino acids are an important aspect of phylogenetic inference, they pose formidable computational challenges and they are usually handled with heuristics [4, 5, 6]. Routine application of approximate inference techniques fails because of the intricate nature of the combinatorial space underlying InDel models.

Concretely, the models considered in the phylogenetic literature take the form of a tree-shaped graphical model where nodes are string-valued random variables representing a fragment of DNA, RNA or protein of a species. Edges denote evolution from one species to another, with conditional probabilities derived from the stochastic model described in Sec. 2. Usually, only the terminal nodes are observed, while the internal nodes are hidden. The interpretation is that the sequence at the root is the common ancestor of those at the terminal nodes, and it subsequently evolved in a branching process following the topology of the tree. We will concentrate on the problem of computing the posterior of these hidden nodes rather than the problem of selecting the topology of the tree—hence we will assume the tree is known or estimated with some other algorithm (a guide tree assumption).

This graphical model can be misleading. It only encodes one type of independence relation, those between generations. There is another important structure that can be exploited. Informally, InDel events that operate at the beginning of the sequences should not affect, for instance, those at the end. However, because alignments between the sequences are unknown in practice, it is difficult to exploit this structure in a principled way.

In many previous works [4, 5, 6], the following heuristic approach is taken to perform inference on the hidden nodes (refer to Fig. 1): First, a guide tree (d) and a *multiple sequence alignment* (a) (a transitive alignment between the characters in the sequences of the modern species) are computed using heuristics [7, 8]. Second, the problem is cast into several easy subproblems as follows. For each equivalence class in the multiple sequence alignment (called a *site*, corresponding to a column in Fig. 1(b)), a new graphical model is created with the same tree structure as the original problem, but where there is exactly one character in each node rather than a string. For nodes with a character

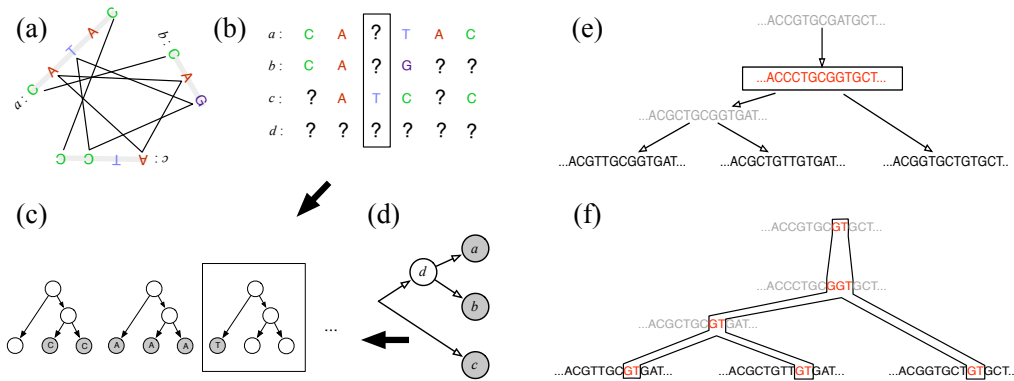

Figure 1: Comparison of different approaches to phylogenetic modeling: (a,b,c,d) heuristics based on site independence; (e) Single Sequence Resampling; (f) Ancestry Resampling. The boxes denote the structures that can be sampled or integrated out in one step by each method.

in the current equivalence class, the node in this new tree is observed, and the rest of the nodes are considered as unobserved data (Fig. 1(c)). Note that the question marks are not the gaps commonly seen in linearized representations of multiple alignments, but rather phantom characters. Finally, each site is assumed independent of the others, so the subproblems can be solved efficiently by running the forward-backward algorithm on each site.

This heuristic has several problems, the most important being that it does not allow explicit modeling of insertions and deletions (InDel), which are frequent in real biological data and play an important role in evolution [9]. If InDels are included in the probabilistic model, there is no longer a deterministic notion of site on which independence assumptions can be made. This complicates inference substantially. For instance, in the standard TKF91 model [10], the fastest known algorithm for computing exact posteriors takes time $O(2^F N^F)$ where $F$ is the number of leaves and $N$ is the geometric mean sequence length [11].

Holmes et al. [2] developed an approximate Markov chain Monte Carlo (MCMC) inference procedure for the TKF91 model. Their algorithm proceeds by sampling the entire sequence corresponding to a single species conditioning on its parent and children (Fig. 1(e)). We will call this type of kernel a *Single Sequence Resampling* (SSR) move. Unfortunately, chains based exclusively on SSR have performance problems.

There are two factors behind these problems. The first factor is a random walk behavior that arises in tall chains found in large or unbalanced trees [2, 12]: initially, the InDel events resampled at the top of the tree are *independent* of all the observations. It takes time for the information from the observations to propagate up the tree. The second factor is the computational cost of each SSR move, which is $O(N^3)$ with the TKF91 model and binary trees. For long sequences, this becomes prohibitive, so it is common to use a "maximum deviation pruning strategy" (i.e., putting a bound on the relative positions of characters that mutate from one to the other) to speed things up [12]. We observed that this pruning can substantially hurt the quality of the estimated posterior (see Sec. 4).

In this paper, we present a novel MCMC procedure for phylogenetic InDel models that we refer to as *Ancestry Resampling* (AR). AR addresses both of the efficiency and accuracy problems that arise for SSR. The intuition behind the AR approach is to use an MCMC kernel that combines the advantages of the two approaches described above: like the forward-backward algorithm in the site-independent case, AR always directly conditions on some part of the observed data, but, like SSR, it is capable of resampling the InDel history. This is illustrated in Fig. 1(f).

## 2   Model

For concreteness, we describe the algorithms in the context of the standard TKF91 model [10], but in Sec. 5 we discuss how the ideas extend to other models. We assume that a phylogenetic directed tree topology $\tau = (V, E)$ is fixed, where nodes in this tree are string-valued random variables, from

an alphabet of $K$ characters—$K$ is four in nucleotide sequences and about twenty in amino-acid sequences. Also known is a positive time length $t_e$ associated to each edge $e \in E$.

We start the description of the model in the simple case of a single branch of known length $t$, with a string $\boldsymbol{x}$ at the root and a string $\boldsymbol{y}$ at the leaf. The model, TKF91, is a string-valued Continuous-Time Markov Chain (CTMC). There is one rate $\mu$ for deletion (death in the original TKF terminology) and one rate $\lambda$ for insertions, which can occur either to the right of one of the existing character (birth), or to the left of the sequence (immigration). Additionally, there is an independent CTMC substitution process on each character.

Fortunately, the TKF91 model has a closed form solution for the conditional distribution over strings $\boldsymbol{y}$ at the leaf given the string $\boldsymbol{x}$ at the root. The derivation of this conditional distribution is presented in [10] and its form is:

$$\mathbb{P}(\text{a character in } \boldsymbol{x} \text{ survived and has } n \text{ descendants in } \boldsymbol{y}) = \alpha\beta^{n-1}(1-\beta) \qquad \text{for } n = 1, 2, \ldots$$
$$\mathbb{P}(\text{a character in } \boldsymbol{x} \text{ died and has } n \text{ descendants in } \boldsymbol{y}) = (1-\alpha)(1-\gamma) \qquad \text{for } n = 0$$
$$= (1-\alpha)\gamma\beta^{n-1}(1-\beta) \quad \text{for } n = 1, 2, \ldots$$
$$\mathbb{P}(\text{immigrants inserted at the left have } n \text{ descendants in } \boldsymbol{y}) = \beta^n(1-\beta) \qquad \text{for } n = 0, 1, \ldots$$

In defining descendants, we count the character itself, its children, grandchildren, etc. $\alpha, \beta, \gamma$ are functions of $t, \mu, \lambda$. See [2] for the details. Since we only work with these conditionals, note that the situation resembles that of a standard weighted edit process with a specific, branch-length dependent structure over insertions and deletions.

To go from a single branch to a tree, we simply compose this process. The full generative process works as follows: starting at the root, we generate the first string according to the stationary distribution of TKF91. Then, for each outgoing edge $e$, we use the known time $t_e$ and the equations above to generate a child string. We continue in preorder recursively.

## 2.1 Auxiliary variables

We now define some auxiliary variables that will be useful in the next section. Between each pair of nodes $a, b \in V$ connected by an edge and with respective strings $\boldsymbol{x}, \boldsymbol{y}$, we define an *alignment* random variable: its values are bipartite matchings between the characters of the strings $\boldsymbol{x}$ and $\boldsymbol{y}$. Links in this alignment denote survival of a character (allowing zero or more substitutions). Note that this alignment is monotonic: if character $i$ in $\boldsymbol{x}$ is linked to character $j$ in $\boldsymbol{y}$, then the characters $i' > i$ in $\boldsymbol{x}$ can only be unlinked or linked to a character with index $j' > j$ in $\boldsymbol{y}$. The random variable that consists of the alignments and the strings for all the edges and nodes in the phylogenetic tree $\tau$ will be called a *derivation*.

Note also that a derivation $D$ defines another graph that we will call a *derivation graph*. Its nodes are the characters of all the strings in the tree. We put an edge between two characters $x, y$ in this graph iff two properties hold. Let $a, b \in V$ be the nodes corresponding to the strings from which respectively $x, y$ belongs to. We put an edge between $x, y$ iff (1) there is an edge between $a$ and $b$ in $E$ and (2) there is a link between $x, y$ in the alignment of the corresponding strings. Examples of derivation graphs are shown in Fig. 2.

## 3 Efficient inference

The approximate inference algorithm we propose, Ancestry Resampling (AR), is based on the Metropolis-Hastings (MH) framework. While the SSR kernel resamples the whole sequence corresponding to a single node, AR works around the difficulties of SSR by joint resampling of a "thin vertical slice" (Fig. 1(f)) in the tree that is composed of a short substring in every node. As we will see, if we use the right definition of vertical slice, this yields a valid and efficient MH algorithm.

## 3.1 Ancestry Resampling

We will call one of these "thin slices" an *ancestry* $\mathcal{A}$, and we now discuss what its definition should be. Some care will be needed to ensure irreducibility and reversibility of the sampler.

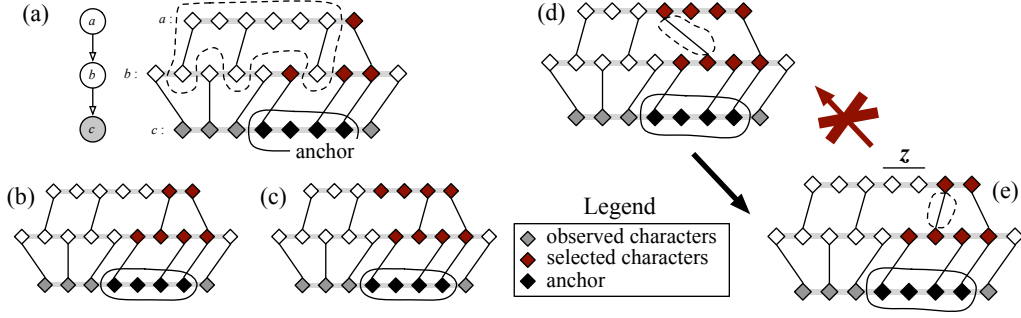

Figure 2: (a): the simple guide tree used in this example (left) and the corresponding sequences and alignments (right). (a,b,c): the definitions of $\mathcal{A}_0, \mathcal{A}_\infty, \mathcal{A}$ respectively are shaded (the "selected characters"). (d,e): An example showing the non-reversibility problem with $\mathcal{A}_\infty$.

We first augment the state of the AR sampler to include the derivation auxiliary variable described in Sec. 2.1. Let $D$ be the current derivation and let $x$ be a substring of one of the terminal nodes, say in node $e$. We will call $x$ an *anchor*. The ancestry will depend on both a derivation and an anchor. The overall MH sampler is a mixture of proposal distributions indexed by a set of anchors covering all the characters in the terminal strings. Each proposal resamples a new value of $\mathcal{A}(D, x)$ given the terminal nodes and keeping $\mathcal{A}(D, x)^c$ frozen.

We first let $\mathcal{A}_0(D, x)$ be the set of characters connected to some character in $x$ in the derivation graph of $D$ (see Fig. 2(a)). This set $\mathcal{A}_0(D, x)$ is not a suitable definition of vertical slice, but will be useful to construct the correct one. It is unsuitable for two reasons. First, it does not yield an irreducible chain, as illustrated in same figure, where nine of the characters of this sample (those inside the dashed curve) will never be resampled, no matter which substring of the terminal node is selected as anchor. Secondly, we would like the vertical slices to be contiguous substrings rather than general subsequences to ease implementation.

We therefore modify the definition recursively as follows. See Fig. 2(b) for an illustration of this definition. For $i > 0$, we will say that a character token $y$ is in $\mathcal{A}_i(D, x)$ if one of the following conditions is true:

1. $y$ is connected to $\mathcal{A}_{i-1}(D, x)$,
2. $y$ appears in a string $\cdots y' \cdots y \cdots y'' \cdots$ such that both $y'$ and $y''$ are in $\mathcal{A}_{i-1}(D, x)$,
3. $y$ appears in a string $\cdots y' \cdots y \cdots$ such that $y'$ is in $\mathcal{A}_{i-1}(D, x)$ and $x$ is a suffix,
4. $y$ appears in a string $\cdots y \cdots y' \cdots$ such that $y'$ is in $\mathcal{A}_{i-1}(D, x)$ and $x$ is a prefix.

Then, we define $\mathcal{A}_\infty(D, x) := \cup_{i=0}^\infty \mathcal{A}_i(D, x)$. In words, a symbol is in $\mathcal{A}_\infty(D, x)$ if it is linked to an anchored character through the alignments, or if it is "squeezed" between previously connected characters. Cases 3 and 4 handle the boundaries of strings. With this property, irreducibility could be established with some conditions on the anchors, but it turns out that this definition is still not quite right.

With $\mathcal{A}_\infty$, the main problem arises when one tries to establish reversibility of the chain. This is illustrated in Fig. 2(d). In this example, the chain first transitions to a new state by altering the circled link. One can see that with the definition of $\mathcal{A}_\infty(D, x)$ given above, from the state 2 (e), the state in 2 (d) is now unreachable by the same resampling operator, the reason being that the substring labeled $z$ in the figure belongs to the frozen part of the state if the transition is visited backwards.

While there exist MCMC methods that are not based on reversible chains [13], we prefer to take a simpler approach: a variation on our definition solves the issue, informally by taking vertical slices $\mathcal{A}(D, x)$ to be the "complement of the ancestry taken on the complement of the anchor." More precisely, if $x = x'xx''$ is the string at the anchor node $e$, we let the resampled section to be $\mathcal{A}(D, x) := (\mathcal{A}_\infty(D, x') \cup \mathcal{A}_\infty(D, x''))^c$. This creates slightly thicker slices (Fig. 2(c)) but solves the reversibility problem. We will call $\mathcal{A}(D, x)$ the *ancestry* of the anchor $x$. With this definition, the proposal distribution can be made reversible using a MH acceptance ratio; it is also irreducible.

The problem of resampling a single slice decomposes along the tree structure $\tau$, but an unbounded number of InDels could occur a priori inside the thin slice. It may seem at the first glance that we are back at our initial problem: sampling from a tree-structured directed graphical model where the support of the space of the nodes is a countably infinite space. But in fact, we have made progress: the distribution is now concentrated on very short sequences. Indeed, the anchors $x$ can be taken relatively small (we used anchors of length 3 to 5 in our experiments).

Another important property to notice is that given an assignment of the random variable $\mathcal{A}(D, x)$, it is possible to compute efficiently and exactly an unnormalized probability for this assignment. The summation over the possible alignments can be done using a standard quadratic dynamic program known in its max version as the Needleman-Wunsch algorithm [14].

## 3.2 Cylindric proposal

We now introduce the second idea that will make efficient inference possible: when resampling an ancestry given its complement, rather than allowing all possible strings for the resampled value of $\mathcal{A}(D, x)$, we restrict the choices to the set of substitutes that are close to its current value. We formalize closeness as follows: Let $\mathfrak{a}_1, \mathfrak{a}_2$ be two values for the ancestry $\mathcal{A}(D, x)$. We define the *cylindric distance* as the maximum over all the nodes $e$ of the Levenshtein edit distance between the substrings in $\mathfrak{a}_1$ and $\mathfrak{a}_2$ at node $e$. Fix some positive integer $m$. The proposal distribution consider the substitution ancestry that are within a ball of radius $m$ centered at the current state in the cylindric metric. The value $m = 1$ worked well in practice.

Here the number of states in the tree-structured dynamic program at each node is polynomial in the lengths of the strings in the current ancestry. A sample can therefore be obtained easily using the observation we have made that unnormalized probability can be computed.[1] Next, we compute the acceptance ratio, i.e.:

$$\min \left\{ 1, \frac{\mathbb{P}(\mathfrak{a}_p) \times Q(\mathfrak{a}_c | \mathfrak{a}_p)}{\mathbb{P}(\mathfrak{a}_c) \times Q(\mathfrak{a}_p | \mathfrak{a}_c)} \right\},$$

where $\mathfrak{a}_c, \mathfrak{a}_p$ are the current and proposed ancestry values and $Q(\mathfrak{a}_2 | \mathfrak{a}_1)$ is the transition probability of the MH kernel, proportional to $\mathbb{P}(\cdot)$, but with support restricted to the cylindric ball centered at $\mathfrak{a}_1$.

# 4 Experiments

We consider two tasks: reconstruction of ancestral sequences and prediction of alignments between multiple genetically-related proteins. We are interested in comparing the ancestry sampling method (AR) presented in this paper with the Markov kernel used in previous literature (SSR).

## 4.1 Reconstruction of ancestral sequences

Given a set of genetically-related sequences, the reconstruction task is to infer properties of the common ancestor of these modern species. This task has important scientific applications: for instance, in [1], the ratio of G+C nucleotide content of ribosomal RNA sequences was estimated to assess the environmental temperature of the common ancestor to all life forms (this ratio is strongly correlated with the optimal growth temperature of prokaryotes).

Just as in the task of topology reconstruction, there are no gold ancestral sequences available to evaluate ancestral sequence reconstruction. For this reason, we take the same approach as in topology reconstruction and perform comparisons on synthetic data [15].

We generated a root node from the DNA alphabet and evolved it down a binary tree of seven nodes. Only the leaves were given to the algorithms (a total of 124010 nucleotides); the hidden nodes were held out. Since our goal in this experiment is to compare inference algorithms rather than methods

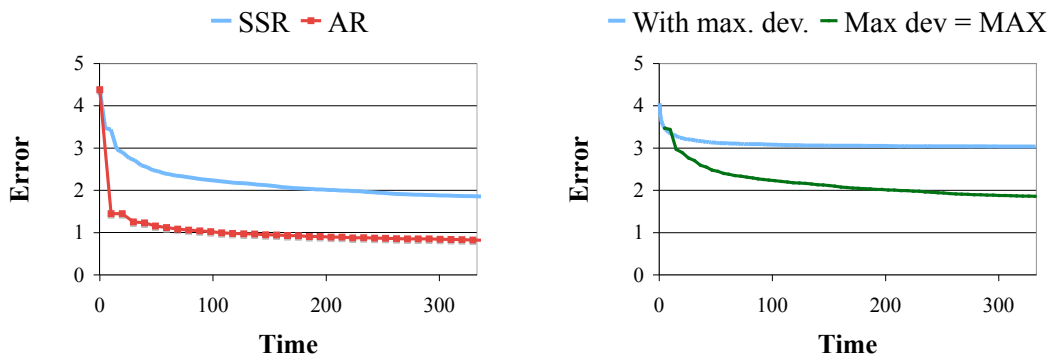

Figure 3: Left: Single Sequence Resampling versus Ancestry Resampling on the sequence reconstruction task. Right: Detrimental effect of a maximum deviation heuristic, which is not needed with AR samplers.

of estimation, we gave both algorithms the true parameters; i.e., those that were used to generate the data.

The task is to predict the sequences at the root node with error measured using the Levenshtein edit distance $l$. For both algorithms, we used a standard approximation to minimum Bayes risk decoding to produce the final reconstruction. If $s_1, s_2, \ldots, s_I$ are the samples collected up to iteration $I$, we return $\min_{i \in 1 \ldots I} \sum_{j \in 1 \ldots I} l(s_i, s_j)$.

Fig. 3 (left) shows the error as a function of time for the two algorithms, both implemented efficiently in Java. Although the computational cost for one pass through the data was higher with AR, the AR method proved to be dramatically more effective: after only one pass through the data (345s), AR already performed better than running SSR for nine hours. Moreover, AR steadily improved its performance as more samples were collected, keeping its error at each iteration to less than half of that of the competitor.

Fig. 3 (right) shows the detrimental effect of a maximum deviation heuristic. This experiment was performed under the same setup described in this section. While the maximum deviation heuristic is necessary for SSR to be able to handle the long sequences found in biological datasets, it is not necessary for AR samplers.

## 4.2 Protein multiple sequence alignment

We also performed experiments on the task of protein multiple sequence alignment, for which the BAliBASE [16] dataset provides a standard benchmark. BAliBASE contains annotations created by biologists using secondary structure elements and other biological cues.

Note first that we can get a multiple sequence alignment from an InDel evolutionary model. For a set $S$ of sequences to align, construct a phylogenetic tree such that its terminal leaves coincide with $S$. A multiple sequence alignment can be extracted from the inferred derivation $D$ as follows: deem the amino acids $x, y \in S$ aligned iff $y \in \mathcal{A}_0(D, x)$.

The state-of-the-art for multiple sequence alignment systems based on an evolutionary model is Handel [2]. It is based on TKF91 and produces a multiple sequence alignment as described above. The key difference with our approach is that their inference algorithm is based on SSR rather than the AR move that we advocate in this paper.

While other heuristic approaches are known to perform better than Handel on this dataset [8, 17], they are not based on explicit evolutionary models. They perform better because they leverage more sophisticated features such as affine gap penalties and hydrophobic core modeling. While these features can be incorporated in our model, we leave this for future work since the topic of this paper is inference.

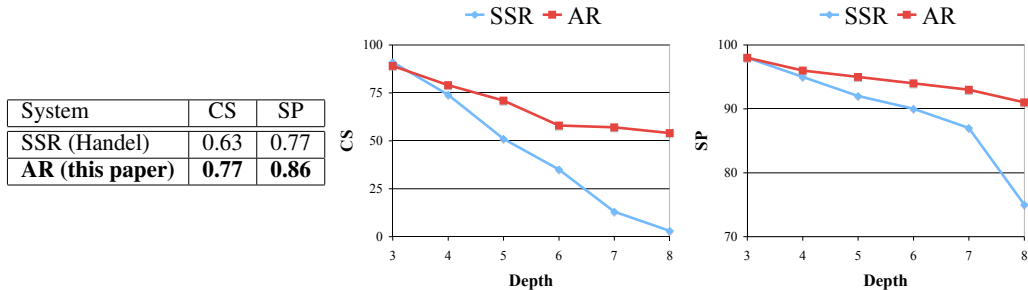

| System | CS | SP |
|---|---|---|
| SSR (Handel) | 0.63 | 0.77 |
| **AR (this paper)** | **0.77** | **0.86** |

Figure 4: Left: performance on the *ref1* directory of BAliBASE. Center, right: Column-Score (CS) and Sum-of-Pairs score (SP) as a function of the depth of the generating trees.

We built evolutionary trees using weighbor [7]. We ran each system for the same time on the sequences in the *ref1* directory of BAliBASE v.1. Decoding for this experiment was done by picking the sample with highest likelihood. We report in Fig. 4(left) the CS and SP Scores, the two standard metrics for this task. Both are recall measures on the subset of the alignments that were labeled, called the core blocks; see, e.g., [17] for instance for the details. For both metrics, our approach performs better.

In order to investigate where the advantage comes from, we did another multiple alignment experiment, plotting performance as a function of the depth of the trees. If the random walk argument presented in the introduction holds, we would expect the advantage of AR over SSR to increase as the tree gets taller. This prediction is confirmed as illustrated in Fig. 4 (center, right). For short trees, the two algorithms perform equally, SSR beating AR slightly for trees with three nodes, which is not surprising since SSR actually performs exact inference in this tiny topology. However, as the trees get taller, the task becomes more difficult, and only AR maintains good performance.

## 5 Conclusion

We have described a principled inference procedure for InDel trees. We have evaluated its performance against a state-of-the-art statistical alignment procedure and shown its clear superiority. In contrast to heuristics such as Clustalw [8], it can be used both for reconstruction of ancestral sequences and multiple alignment.

While our algorithm was described in the context of TKF91, it can be extended to more sophisticated models. Incorporating affine gap penalties and hydrophobic core modeling is of particular interest as they are known to dramatically improve multiple alignment performance [2]. These models typically do not have closed forms for the conditional probabilities, but this could be alleviated by using a discretization of longer branches. This creates tall trees, but as we have seen, AR still performs very well in this setting.

## Footnotes

[1]What we are using here is actually a *nested dynamic programs*, meaning that the computation of a probability in the outer dynamic program (DP) requires the computation of an inner, simpler DP. While this may seem prohibitive, this is made feasible by designing the sampling kernels so that the inner DP is executed most of the time on small problem instances. We also cached the small-DP cost matrices.

## References

[1] N. Galtier, N. Tourasse, and M. Gouy. A nonhyperthermophilic common ancestor to extant life forms. *Science*, 283:220–221, 1999.

[2] I. Holmes and W. J. Bruno. Evolutionary HMM: a Bayesian approach to multiple alignment. *Bioinformatics*, 17:803–820, 2001.

[3] J. Felsenstein. *Inferring Phylogenies*. Sinauer Associates, 2003.

[4] Z. Yang and B. Rannala. Bayesian phylogenetic inference using DNA sequences: A Markov chain Monte Carlo method. *Molecular Biology and Evolultion*, 14:717–724, 1997.

[5] B. Mau and M. A. Newton. Phylogenetic inference for binary data on dendrograms using Markov chain Monte Carlo. *Journal of Computational and Graphical Statistics*, 6:122–131, 1997.

[6] S. Li, D. K. Pearl, and H. Doss. Phylogenetic tree construction using Markov chain Monte Carlo. *Journal of the American Statistical Association*, 95:493–508, 2000.

[7] W. J. Bruno, N. D. Socci, and A. L. Halpern. Weighted neighbor joining: A likelihood-based approach to distance-based phylogeny reconstruction. *Molecular Biology and Evolution*, 17:189–197, 2000.

[8] D. G. Higgins and P. M. Sharp. CLUSTAL: a package for performing multiple sequence alignment on a microcomputer. *Gene*, 73:237–244, 1988.

[9] J. L. Thorne, H. Kishino, and J. Felsenstein. Inching toward reality: an improved likelihood model of sequence evolution. *Journal of Molecular Evolution*, 34:3–16, 1992.

[10] J. L. Thorne, H. Kishino, and J. Felsenstein. An evolutionary model for maximum likelihood alignment of DNA sequences. *Journal of Molecular Evolution*, 33:114–124, 1991.

[11] G. A. Lunter, I. Miklós, Y. S. Song, and J. Hein. An efficient algorithm for statistical multiple alignment on arbitrary phylogenetic trees. *Journal of Computational Biology*, 10:869–889, 2003.

[12] A. Bouchard-Côté, P. Liang, D. Klein, and T. L. Griffiths. A probabilistic approach to diachronic phonology. In *Proceedings of EMNLP 2007*, 2007.

[13] P. Diaconis, S. Holmes, and R. M. Neal. Analysis of a non-reversible Markov chain sampler. Technical report, Cornell University, 1997.

[14] S. Needleman and C. Wunsch. A general method applicable to the search for similarities in the amino acid sequence of two proteins. *Journal of Molecular Biology*, 48:443–453, 1970.

[15] K. St. John, T. Warnow, B. M. E. Moret, and L. Vawter. Performance study of phylogenetic methods: (unweighted) quartet methods and neighbor-joining. *Journal of Algorithms*, 48:173–193, 2003.

[16] J. Thompson, F. Plewniak, and O. Poch. BAliBASE: A benchmark alignments database for the evaluation of multiple sequence alignment programs. *Bioinformatics*, 15:87–88, 1999.

[17] C. B. Do, M. S. P. Mahabhashyam, M. Brudno, and S. Batzoglou. PROBCONS: Probabilistic consistency-based multiple sequence alignment. *Genome Research*, 15:330–340, 2005.

